# Recovery of Sparse Probability Measures via Convex Programming

**Mert Pilanci and Laurent El Ghaoui**
Electrical Engineering and Computer Science
University of California Berkeley
Berkeley, CA 94720
{mert,elghaoui}@eecs.berkeley.edu

**Venkat Chandrasekaran**
Department of Computing and Mathematical Sciences
California Institute of Technology
Pasadena, CA 91125
venkatc@caltech.edu

## Abstract

We consider the problem of cardinality penalized optimization of a convex function over the probability simplex with additional convex constraints. The classical $\ell_1$ regularizer fails to promote sparsity on the probability simplex since $\ell_1$ norm on the probability simplex is trivially constant. We propose a direct relaxation of the minimum cardinality problem and show that it can be efficiently solved using convex programming. As a first application we consider recovering a sparse probability measure given moment constraints, in which our formulation becomes linear programming, hence can be solved very efficiently. A sufficient condition for exact recovery of the minimum cardinality solution is derived for arbitrary affine constraints. We then develop a penalized version for the noisy setting which can be solved using second order cone programs. The proposed method outperforms known rescaling heuristics based on $\ell_1$ norm. As a second application we consider convex clustering using a sparse Gaussian mixture and compare our results with the well known soft k-means algorithm.

## 1 Introduction

We consider optimization problems of the following form,

$$p^* = \min_{x \in C,\ 1^T x = 1,\ x \geq 0} f(x) + \lambda \mathbf{card}(x)$$

where $f$ is a convex function, $C$ is a convex set, $\mathbf{card}(x)$ denotes the number of nonzero elements of x and $\lambda \geq 0$ is a given tradeoff parameter for adjusting desired sparsity. Since the cardinality penalty is inherently of combinatorial nature, these problems are in general not solvable in polynomial-time. In recent years $\ell_1$ norm penalization as a proxy for penalizing cardinality has attracted a great deal of attention in machine learning, statistics, engineering and applied mathematics [1], [2], [3], [4]. However the aforementioned types of sparse probability optimization problems are not amenable to the $\ell_1$ *heuristic* since $\|x\|_1 = 1^T x = 1$ is constant on the probability simplex. Numerous problems in machine learning, statistics, finance and signal processing fall into this category however to the authors' knowledge there is no known general convex optimization strategy for such problems constrained on the probability simplex. The aim of this paper is to claim that the reciprocal of the

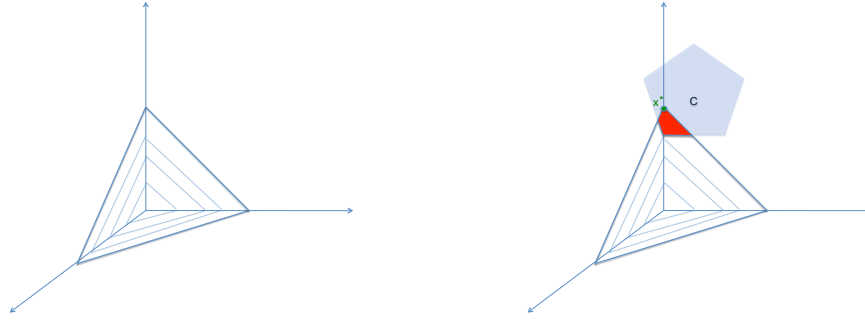

(a) Level sets of the regularization function $\frac{1}{\max_i x_i}$ on the probability simplex

(b) The sparsest probability distribution on the set $C$ is $x^*$ (green) which also minimizes $\frac{1}{\max_i x_i}$ on the intersection (red)

Figure 1: Probability simplex and the reciprocal of the infinity norm

infinity-norm, i.e., $\frac{1}{\max_i x_i}$ can be used as a convex heuristic for penalizing cardinality on the probability simplex and the resulting relaxations can be solved via convex optimization. Figure 1(a) and 1(b) depict the level sets and an example of a sparse probability measure which has maximal infinity norm. In the following sections we expand our discussion by exploring two specific problems: recovering a measure from given moments where $f = 0$ and $C$ is affine, and convex clustering where $f$ is a log-likelihood and $C = \mathbb{R}$. For the former case we give a sufficient condition for this convex relaxation to exactly recover the minimal cardinality solution of $p^*$. We then present numerical simulations for the both problems which suggest that the proposed scheme offers a very efficient convex relaxation for penalizing cardinality on the probability simplex.

## 2 Optimizing over sparse probability measures

We begin the discussion by first taking an alternative approach to the cardinality penalized optimization by directly lower-bounding the original hard problem using the following relation

$$\|x\|_1 = \sum_{i=1}^{n} |x_i| \leq \mathbf{card}(x) \max_i |x_i| \leq \mathbf{card}(x) \|x\|_\infty$$

which is essentially one of the core motivations of using $\ell_1$ penalty as a proxy for cardinality. When constrained to the probability simplex, the lower-bound for the cardinality simply becomes $\frac{1}{\max_i x_i} \leq \mathbf{card}(x)$. Using this bound on the cardinality, we immediately have a lower-bound on our original NP-hard problem which we denote by $p_\infty^*$:

$$p^* \geq p_\infty^* := \min_{x \in C, \ 1^T x = 1, \ x \geq 0} f(x) + \lambda \frac{1}{\max_i x_i} \tag{1}$$

The function $\frac{1}{\max_i x_i}$ is concave and hence the above lower-bounding problem is not a convex optimization problem. However below we show that the above problem can be exactly solved using convex programming.

**Proposition 2.1.** *The lower-bounding problem defined by $p_\infty^*$ can be globally solved using the following $n$ convex programs in $n + 1$ dimensions:*

$$p^* \geq p_\infty^* = \min_{i=1,\ldots,n} \left\{ \min_{x \in C, \ 1^T x = 1, \ x \geq 0, \ t \geq 0} f(x) + t \ : \ x_i \geq \lambda/t \right\}. \tag{2}$$

Note that the constraint $x_i \geq \lambda/t$ is jointly convex since $1/t$ is convex in $t \in \mathbb{R}^+$, and they can be handled in most of the general purpose convex optimizers, e.g. cvx, using either the positive inverse function or rotated cone constraints.

*Proof.*

$$p_\infty^* = \min_{x \in C,\ 1^T x = 1,\ x \geq 0} f(x) + \min_i \frac{\lambda}{x_i} \tag{3}$$

$$= \min_i \min_{x \in C,\ 1^T x = 1,\ x \geq 0} f(x) + \frac{\lambda}{x_i} \tag{4}$$

$$= \min_i \min_{x \in C,\ 1^T x = 1,\ x \geq 0, t \geq 0} f(x) + t \quad s.t. \quad \frac{\lambda}{x_i} \leq t \tag{5}$$

$\square$

The above formulation can be used to efficiently approximate the original cardinality constrained problem by lower-bounding for arbitrary convex $f$ and $C$. In the next section we show how to compute the quality of approximation.

## 2.1 Computing a bound on the quality of approximation

By the virtue of being a relaxation to the original cardinality problem, we have the following remarkable property. Let $\hat{x}$ be an optimal solution to the convex program $p_\infty^*$, then we have the following relation

$$f(\hat{x}) + \lambda \mathbf{card}(\hat{x}) \geq p^* \geq p_\infty^* \tag{6}$$

Since the left-hand side and right-hand side of the above bound are readily available when $p_\infty^*$ defined in (2) is solved, we immediately have a bound on the quality of relaxation. More specifically the relaxation is exact, i.e., we find a solution for the original cardinality penalized problem, if the following holds:

$$f(\hat{x}) + \lambda \mathbf{card}(\hat{x}) = p_\infty^*$$

It should be noted that for general cardinality penalized problems, using $\ell_1$ heuristic does not yield such a quality bound, since it is not a lower or upper bound in general. Moreover most of the known equivalence conditions for $\ell_1$ heuristics such as Restricted Isometry Property and variants are NP-hard to check. Therefore a remarkable property of the proposed scheme is that it comes with a simple computable bound on the quality of approximation.

## 3 Recovering a Sparse Measure

Suppose that $\mu$ is a discrete probability measure and we would like to know the sparsest measure satisfying some arbitrary moment constraints:

$$p^* = \min_\mu \mathbf{card}(\mu) \quad : \quad \mathbb{E}_\mu[X_i] = b_i,\ i = 1, \dots, m$$

where $X_i$'s are random variables and $E_\mu$ denotes expectation with respect to the measure $\mu$. One motivation for the above problem is the fact that it upper-bounds the minimum entropy power problem:

$$p^* \geq \min_\mu \exp H(\mu) \quad : \quad \mathbb{E}_\mu[X_i] = b_i,\ i = 1, \dots, m$$

where $H(\mu) := -\sum_i \mu_i \log \mu_i$ is the Shannon entropy. Both of the above problems are non-convex and in general very hard to solve.

When viewed as a finite dimensional optimization problem the minimum cardinality problem can be cast as a linear sparse recovery problem:

$$p^* = \min_{1^T x = 1,\ x \geq 0} \mathbf{card}(x) \quad : \quad Ax = b \tag{7}$$

As noted previously, applying the $\ell_1$ heuristic doesn't work and it does not even yield a unique solution when the problem is underdetermined since it simply solves a feasibility problem:

$$p_1^* = \min_{1^T x = 1,\ x \geq 0} \|x\|_1 \quad : \quad Ax = b \tag{8}$$

$$= \min_{1^T x = 1,\ x \geq 0} 1 \quad : \quad Ax = b \tag{9}$$

and recovers the true minimum cardinality solution if and only if the set $1^T x = 1, \ x \geq 0, \ Ax = b$ is a singleton. This condition may hold in some cases, i.e. when the first $2k - 1$ moments are available, i.e., $A$ is a Vandermonde matrix where $k = \mathbf{card}(x)$ [6]. However in general this set is a polyhedron containing dense vectors. Below we show how the proposed scheme applies to this problem.

Using general form in (2), the proposed relaxation is given by the following,

$$(p^*)^{-1} \leq (p_\infty^*)^{-1} = \max_{i=1,\ldots,n} \left\{ \max_{1^T x = 1, \ x \geq 0} x_i \ : \ Ax = b \right\}. \tag{10}$$

which can be solved very efficiently by solving $n$ linear programs in $n$ variables. The total complexity is at most $O(n^4)$ using a primal-dual LP solver.

It's easy to check that strong duality holds and the dual problems are given by the following:

$$(p_\infty^*)^{-1} = \max_{i=1,\ldots,n} \left\{ \min_{w, \ \lambda} w^T b + \lambda \ : \ A^T w + \lambda 1 \geq e_i \right\}. \tag{11}$$

where 1 is the all ones vector and $e_i$ is all zeros with a one in only $i$'th coordinate.

## 3.1 An alternative minimal cardinality selection scheme

When the desired criteria is to find a minimum cardinality probability vector satisfying $Ax = b$, the following alternative selection scheme offers a further refinement, by picking the lowest cardinality solution among the $n$ linear programming solutions. Define

$$\hat{x}_i : \ = \ \arg \max_{1^T x = 1, \ x \geq 0} x_i \ : \ Ax = b \tag{12}$$

$$\hat{x}_{\min} : \ = \ \arg \min_{i=1,\ldots,n} \mathbf{card}(\hat{x}_i) \tag{13}$$

The following theorem gives a sufficient condition for the recovery of a sparse measure using the above method.

**Theorem 3.1.** *Assume that the solution to $p^*$ in (7) is unique and given by $x^*$. If the following condition holds*

$$\min_{1^T x = 1, \ y \geq 0, \ 1^T y = 1} x_i \ s.t. \ A_S x = A_{S^c} y \quad > 0$$

*where $b = Ax^*$ and $A_S$ is the submatrix containing columns of $A$ corresponding to non-zero elements of $x*$ and $A_{S^c}$ is the submatrix of remaining columns, then the convex linear program*

$$\max_{1^T x = 1, \ x \geq 0} x_i \ : \ Ax = b$$

*has a unique solution given by $x^*$.*

Let $Conv(a_1, \ldots, a_m)$ denote the convex hull of the $m$ vectors $\{a_1, \ldots, a_m\}$. The following corollary depicts a geometric condition for recovery.

**Corollary 3.2.** *If $Conv(A_{S^c})$ does not intersect an extreme point of $Conv(A_S)$ then $\hat{x}_{\min} = x^*$, i.e. we recover the minimum cardinality solution using $n$ linear programs.*

*Proof Outline:*
Consider $k$'th inner linear program defined in the problem $p_\infty^*$. Using the optimality conditions of the primal-dual linear program pairs in (10) and (11), it can be shown that the existence of a pair $(w, \lambda)$ satisfying

$$A_S^T w + \lambda 1 \ = \ e_k \tag{14}$$
$$A_{S^c}^T w + \lambda 1 \ > \ 0 \tag{15}$$

implies that the support of solution of the linear program is exactly equal to the support of $x^*$, and in particular they have the same cardinality. Since the solution of $p^*$ is unique and has minimum cardinality, we conclude that $x^*$ is indeed the unique solution to the $k$'th linear program. Applying Farkas' lemma and duality theory we arrive at the conditions defined in Theorem 3.1. The corollary follows by first observing that the condition of Theorem 3.1 is satisfied if $Conv(A_{S^c})$ does not intersect an extreme point of $Conv(A_S)$. Finally observe that if any of the $n$ linear programs recover the minimal cardinality solution then $\hat{x}_{\min} = x^*$, since $\mathbf{card}(\hat{x}_{\min}) \leq \mathbf{card}(\hat{x}_k), \forall k$.

## 3.2 Noisy measure recovery

When the data contains noise and inaccuracies, such as the case when using empirical moments instead of exact moments, we propose the following noise-aware robust version, which follows from the general recipe given in the first section:

$$\min_{i=1,\ldots,n} \left\{ \min_{1^T x=1,\ x\geq 0, t\geq 0} \|Ax-b\|_2^2 + t \ :\ x_i \geq \lambda/t \right\}. \tag{16}$$

where $\lambda \geq 0$ is a penalty parameter for encouraging sparsity. The above problem can be solved using $n$ second-order cone programs in $n+1$ variables, hence has $O(n^4)$ worst case complexity.

The proposed measure recovery algorithms are investigated and compared with a known suboptimal heuristic in Section 6.

## 4   Convex Clustering

In this section we base our discussion on the exemplar based convex clustering framework of [8]. Given a set of data points $\{z_1, \ldots, z_n\}$ of $d$-dimensional vectors, the task of clustering is to fit a mixture probability model to maximize the log likelihood function

$$L := \frac{1}{n} \sum_{i=1}^{n} \log \left[ \sum_{j=1}^{k} x_j f(z_i; m_j) \right]$$

where $f(z; m)$ is an exponential family distribution on $Z$ with parameter $m$, and $x$ is a k-dimensional vector on the probability simplex denoting the mixture weights. For the standard multivariate Normal distribution we have $f(z_i; m_j) = e^{-\beta \|z_i - m_j\|_2^2}$ for some parameter $\beta > 0$. As in [8] we'll further assume that the mean parameter $m_j$ is one of the examples $z_i$ which is unknown a-priori. This assumption helps to simply the log-likelihood whose data dependence is now only through a kernel matrix $K_{ij} := e^{-\beta \|z_i - z_j\|_2^2}$ as follows

$$L \quad = \quad \frac{1}{n} \sum_{i=1}^{n} \log \left[ \sum_{j=1}^{k} x_j e^{-\beta \|z_i - z_j\|_2^2} \right] \tag{17}$$

$$= \quad \frac{1}{n} \sum_{i=1}^{n} \log \left[ \sum_{j=1}^{k} x_j K_{ij} \right] \tag{18}$$

Partitioning the data $\{z_1, \ldots, z_n\}$ into few clusters is equivalent to have a sparse mixture $x$, i.e., each example is assigned to few centers (which are some other examples). Therefore to cluster the data we propose to approximate the following cardinality penalized problem,

$$p_c^* := \max_{1^T x=1,\ x\geq 0} \sum_{i=1}^{n} \log \left[ \sum_{j=1}^{k} x_j K_{ij} \right] - \lambda \mathbf{card} x \tag{19}$$

As hinted previously, the above problem can be seen as a lower-bound for the entropy penalized problem

$$p_c^* \leq \max_{1^T x=1,\ x\geq 0} \sum_{i=1}^{n} \log \left[ \sum_{j=1}^{k} x_j K_{ij} \right] - \lambda \exp H(x) \tag{20}$$

where $H(x)$ is the Shannon entropy of the mixture probability vector.

Applying our convexification strategy, we arrive at another upper-bound which can be computed via convex optimization

$$p_c^* \leq p_\infty^* := \max_{1^T x=1,\ x\geq 0} \sum_{i=1}^{n} \log \left[ \sum_{j=1}^{k} x_j K_{ij} \right] - \frac{\lambda}{\max_i x_i} \tag{21}$$

We investigate the above approach in a numerical example in Section 6 and compare with the well-known soft k-means algorithm.

# 5 Algorithms

## 5.1 Exponentiated Gradient

Exponentiated gradient [7] is a proximal algorithm to optimize over the probability simplex which employs the Kullback-Leibler divergence $D(x, y) = \sum_i x_i \log \frac{x_i}{y_i}$ between two probability distributions. For minimizing a convex function $\psi$ the exponentiated gradient updates are given by the following:

$$x^{k+1} = \arg\min_x \ \psi(x^k) + \nabla\psi(x^k)^T(x - x^k) + \frac{1}{\alpha}D(x, x^k)$$

When applied to the general form of 2 it yields the following updates to solve the $i$'th problem of $p_\infty^*$

$$x_i^{k+1} = r_i^k x_i^k / \left( \sum_j r_j^k x_j^k \right)$$

where the weights $r_i$ are exponentiated gradients:

$$r_i^k = \exp\left( \alpha(\nabla_i f(x^k) - \lambda/x_i^2) \right)$$

We also note that the above updates can be done in parallel for the $n$ convex programs, and they are guaranteed to converge to the optimum.

# 6 Numerical Results

## 6.1 Recovering a Measure from Gaussian Measurements

Here we show that the proposed recovery scheme is able to recover a sparse measure exactly with overwhelming probability, when the matrix $A \in \mathbb{R}^{m \times n}$ is chosen from the independent Gaussian ensemble, i.e, $A_{i,j} \sim \mathcal{N}(0, 1)$ i.i.d.

As an alternative method we consider a commonly employed simple heuristic to optimize over a probability measure which first drops the constraint $1^T x = 1$ and solves the corresponding $\ell_1$ penalized problem. And finally rescales the optimal $x$ such that $1^T x = 1$. In the worst case, this procedure recovers the true solution whenever minimizing $\ell_1$-norm recovers the solution, i.e., when there is only one feasible vector satistfying $Ax = b$ and $x \geq 0$, $1^T x = 1$. This is clearly a suboptimal approach and we will refer it as the *rescaling heuristic*. We set $n = 50$ and randomly pick a probability vector $x^*$ which is $k$ sparse, let $b = Ax^*$ be $m$ noiseless measurements, then check the probability of recovery, i.e. $\hat{x} = x^*$ where $\hat{x}$ is the solution to,

$$\max_{i=1,\ldots,n} \ \left\{ \max_{1^T x=1, \ x\geq 0} x_i \ : \ Ax = b \right\}. \tag{22}$$

Figure 2(a) shows the probability of *exact* recovery as a function of $m$, the number of measurements, in 100 independent realizations of $A$ for the proposed LP formulation and the rescaling heuristic. As it can be seen in Figure 2(a), the proposed method recovers the correct measure with probability almost 1 when $m \geq 5$. Quite interestingly the rescaling heuristic doesn't succeed to recover the true measure with high probability even for a cardinality 2 vector.

We then add normal distributed noise with standard deviation $0.1$ on the observations and solve,

$$\min_{i=1,\ldots,n} \ \left\{ \min_{1^T x=1, \ x\geq 0, t\geq 0} \|Ax - b\|_2^2 + t \ : \ x_i \geq \lambda/t \right\}. \tag{23}$$

We compare the above approach by the corresponding rescaling heuristic, which first solves a non-negative Lasso,

$$\min_{x\geq 0} \ \|Ax - b\|_2^2 + \lambda \|x\|_1 \tag{24}$$

then rescales $x$ such that $1^T x = 1$. For each realization of $A$ and measurement noise we run both methods using a primal-dual interior point solver for 30 equally spaced values of $\lambda \in [0, 10]$ and record the minimum error $\|\hat{x} - x^*\|_1$. The average error over 100 realizations are shown in Figure 2(b). Is it can be seen in the figure the proposed scheme clearly outperforms the rescaling heuristic since it can utilize the fact that $x$ is on the probability simplex, without trivializing it's complexity regularizer.

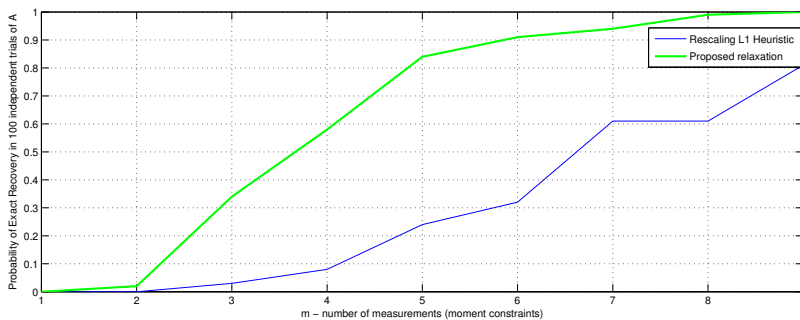

(a) Probability of exact recovery as a function of $m$

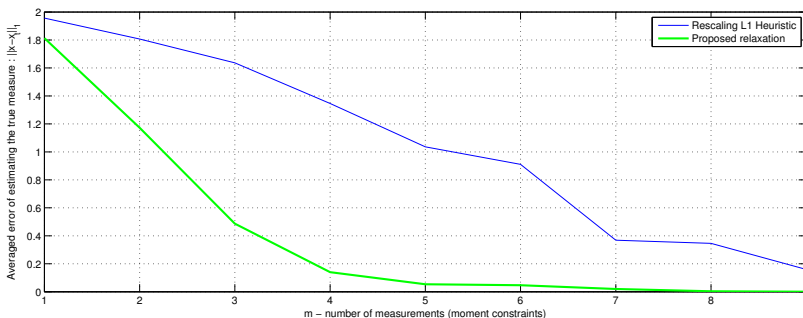

(b) Average error for noisy recovery as a function of $m$

Figure 2: A comparison of the exact recovery probability in the noiseless setting (top) and estimation error in the noisy setting (bottom) of the proposed approach and the rescaled $\ell_1$ heuristic

## 6.2 Convex Clustering

We generate synthetic data using a Gaussian mixture of 10 components with identity covariances and cluster the data using the proposed method, the resulting clusters given by the mixture density is presented in Figure 3. The centers of the circles represent the means of the mixture components and the radii are proportional to the respective mixture weights. We then repeat the clustering procedure using the well known soft k-means algorithm and present the results in Figure 4.

As it can be seen from the figures the proposed convex relaxation is able to penalize the cardinality on the mixture probability vector and produce clusters significantly better than soft k-means algorithm. Note that soft k-means is a non-convex procedure whose performance depends heavily on the initialization. The proposed approach is convex hence insensitive to the initializations. Note that in [8] the number of clusters are adjusted indirectly by varying the $\beta$ parameter of the distribution. In contrast our approach tries to implicitly optimizes the likelihood/cardinality tradeoff by varying $\lambda$. Hence when the number of clusters is unknown, choosing a value of $\lambda$ is usually easier than specificying a value of $k$ for the k-means algorithms.

## 7 Conclusions and Future Directions

We presented a convex cardinality penalization scheme for problems constrained on the probability simplex. We then derived a sufficient condition for recovering the sparsest probability measure in an affine space using the proposed method. The geometric interpretation suggests that it holds for a large class of matrices. An open theoretical question is to analyze the probability of exact recovery for a normally distributed $A$. Another interesting direction is to extend the recovery analysis to the noisy setting and arbitrary functions such as the log-likelihood in the clustering example. There might also be other problems where proposed approach could be practically useful such as portfolio optimization, where a sparse convex combination of assets is sought or sparse multiple kernel learn-

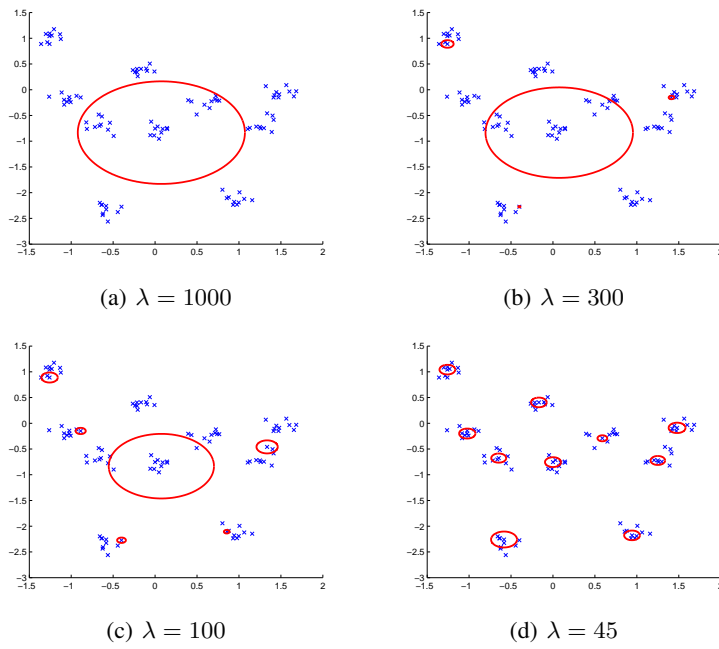

(a) $\lambda = 1000$     (b) $\lambda = 300$

(c) $\lambda = 100$     (d) $\lambda = 45$

Figure 3: Proposed convex clustering scheme

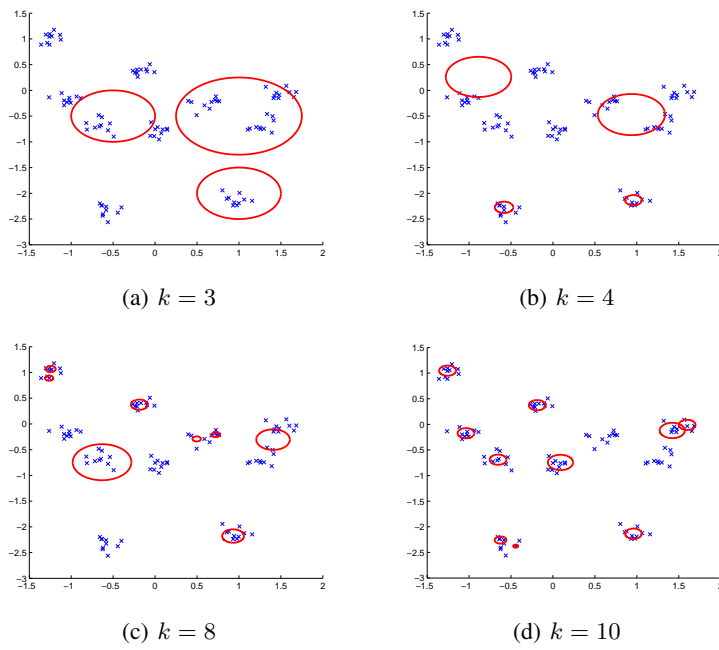

(a) $k = 3$     (b) $k = 4$

(c) $k = 8$     (d) $k = 10$

Figure 4: Soft k-means algorithm

ing.

**Acknowledgements** This work is partially supported by the National Science Foundation under Grants No. CMMI-0969923, FRG-1160319, and SES-0835531, as well as by a University of California CITRIS seed grant, and a NASA grant No. NAS2-03144. The authors would like to thank the Area Editor and the reviewers for their careful review of our submission.

# References

[1] E.J. Candés, T. Tao, *"Decoding by linear programming"*. IEEE Trans. Inform. Theory 51 (2005), 4203-4215.

[2] S. Chen, D. Donoho, and M. Saunders. *Atomic decomposition by basis pursuit"* SIAM Review, 43(1):129-159, 2001.

[3] A. Bruckstein, D. Donoho, and M. Elad. *"From sparse solutions of systems of equations to sparse modeling of signals and images"*. SIAM Review, 2007.

[4] V. Chandrasekaran, B. Recht, P.A. Parrilo, and A.S. Willsky. *"The convex algebraic geometry of linear inverse problems"*. In Communication, Control, and Computing (Allerton), 2010 48th Annual Allerton Conference on, pages 699-703, 2010.

[5] S. Boyd and L. Vandenberghe, *"Convex Optimization"*. Cambridge, U.K.: Cambridge Univ. Press, 2003.

[6] A. Cohen and A. Yeredor, *"On the use of sparsity for recovering discrete probability distributions from their moments"*. Statistical Signal Processing Workshop (SSP), 2011 IEEE

[7] J. Kivinen and M. Warmuth. *"Exponentiated gradient versus gradient descent for linear predictors"*. Information and Computation, 132(1):1-63, 1997.

[8] D. Lashkari and P. Golland, *"Convex clustering with exemplar-based models"*, in *NIPS*, 2008.
